# Constrained Independent Component Analysis

**Wei Lu and Jagath C. Rajapakse**

School of Computer Engineering
Nanyang Technological University, Singapore 639798
email: *asjagath@ntu.edu.sg*

## Abstract

The paper presents a novel technique of constrained independent component analysis (CICA) to introduce constraints into the classical ICA and solve the constrained optimization problem by using Lagrange multiplier methods. This paper shows that CICA can be used to order the resulted independent components in a specific manner and normalize the demixing matrix in the signal separation procedure. It can systematically eliminate the ICA's indeterminacy on permutation and dilation. The experiments demonstrate the use of CICA in ordering of independent components while providing normalized demixing processes.

*Keywords:* Independent component analysis, constrained independent component analysis, constrained optimization, Lagrange multiplier methods

## 1 Introduction

Independent component analysis (ICA) is a technique to transform a multivariate random signal into a signal with components that are mutually independent in complete statistical sense [1]. There has been a growing interest in research for efficient realization of ICA neural networks (ICNNs). These neural algorithms provide adaptive solutions to satisfy independent conditions after the convergence of learning [2, 3, 4].

However, ICA only defines the directions of independent components. The magnitudes of independent components and the norms of demixing matrix may still be varied. Also the order of the resulted components is arbitrary. In general, ICA has such an inherent indeterminacy on dilation and permutation. Such indetermination cannot be reduced further without additional assumptions and constraints [5]. Therefore, constrained independent component analysis (CICA) is proposed as a way to provide a unique ICA solution with certain characteristics on the output by introducing constraints:

- To avoid the arbitrary ordering on output components: statistical measures give indices to sort them in order, and evenly highlight the salient signals.

- To produce unity transform operators: normalization of the demixing channels reduces dilation effect on resulted components. It may recover the exact original sources.

With such conditions applied, the ICA problem becomes a constrained optimization problem. In the present paper, Lagrange multiplier methods are adopted to provide an adaptive solution to this problem. It can be well implemented as an iterative updating system of neural networks, referred to ICNNs. Next section briefly gives an introduction to the problem, analysis and solution of Lagrange multiplier methods. Then the basic concept of ICA will be stated. And Lagrange multiplier methods are utilized to develop a systematic approach to CICA. Simulations are performed to demonstrate the usefulness of the analytical results and indicate the improvements due to the constraints.

## 2  Lagrange Multiplier Methods

Lagrange multiplier methods introduce Lagrange multipliers to resolve a constrained optimization iteratively. A penalty parameter is also introduced to fit the condition so that the local convexity assumption holds at the solution. Lagrange multiplier methods can handle problems with both equality and inequality constraints.

The *constrained nonlinear optimization problems* that Lagrange multiplier methods deal take the following general form:

$$\text{minimize } f(\mathbf{X}), \quad \text{subject to } \mathbf{g}(\mathbf{X}) \leq 0, \ \mathbf{h}(\mathbf{X}) = 0 \tag{1}$$

where $\mathbf{X}$ is a matrix or a vector of the problem arguments, $f(\mathbf{X})$ is an objective function, $\mathbf{g}(\mathbf{X}) = [g_1(\mathbf{X}) \cdots g_m(\mathbf{X})]^{\mathrm{T}}$ defines a set of $m$ inequality constraints and $\mathbf{h}(\mathbf{X}) = [h_1(\mathbf{X}) \cdots h_n(\mathbf{X})]^{\mathrm{T}}$ defines a set of $n$ equality constraints. Because Lagrangian methods cannot directly deal with inequality constraints $g_i(\mathbf{X}) \leq 0$, it is possible to transform inequality constraints into equality constraints by introducing a vector of slack variables $\mathbf{z} = [z_1 \cdots z_m]^{\mathrm{T}}$ to result in equality constraints $p_i(\mathbf{X}) = g_i(\mathbf{X}) + z_i^2 = 0, \ i = 1 \cdots m$.

Based on the transformation, the corresponding simplified augmented Lagrangian function for problem (1) is defined as:

$$\mathcal{L}(\mathbf{X}, \mu, \lambda) \ = \ f(\mathbf{X}) + \frac{1}{2\gamma} \sum_{i=1}^{m} \{[\max\{0, \bar{g}_i(\mathbf{X})\}]^2 - \mu_i^2\} + \lambda^{\mathrm{T}} \mathbf{h}(\mathbf{X}) + \frac{1}{2}\gamma \|\mathbf{h}(\mathbf{X})\|^2 \tag{2}$$

where $\mu = [\mu_1 \cdots \mu_m]^{\mathrm{T}}$ and $\lambda = [\lambda_1 \cdots \lambda_n]^{\mathrm{T}}$ are two sets of Lagrange multipliers, $\gamma$ is the scalar penalty parameter, $\bar{g}_i(\mathbf{X})$ equals to $\mu_i + \gamma g_i(\mathbf{X})$, $\|\cdot\|$ denotes Euclidean norm, and $\frac{1}{2}\gamma \|\cdot\|^2$ is the penalty term to ensure that the optimization problem is held at the condition of local convexity assumption: $\nabla_{\mathbf{XX}}^2 \mathcal{L} > 0$. We use the augmented Lagrangian function in this paper because it gives wider applicability and provides better stability [6].

For discrete problems, the changes in the augmented Lagrangian function can be defined as $\Delta_{\mathbf{X}} \mathcal{L}(\mathbf{X}, \mu, \lambda)$ to achieve the saddle point in the discrete variable space. The iterative equations to solve the problem in eq.(2) are given as follows:

$$\begin{aligned} \mathbf{X}(k+1) \ &= \ \mathbf{X}(k) - \Delta_{\mathbf{X}} \mathcal{L}(\mathbf{X}(k), \mu(k), \lambda(k)) \\ \mu(k+1) \ &= \ \mu(k) + \gamma \, \mathbf{p}(\mathbf{X}(k)) = \max\{0, \bar{\mathbf{g}}(\mathbf{X}(k))\} \\ \lambda(k+1) \ &= \ \lambda(k) + \gamma \, \mathbf{h}(\mathbf{X}(k)) \end{aligned} \tag{3}$$

where $k$ denotes the iterative index and $\bar{\mathbf{g}}(\mathbf{X}(k)) = \mu(k) + \gamma \, \mathbf{g}(\mathbf{X}(k))$.

## 3 Unconstrained ICA

Let the time varying input signal be $\mathbf{x} = (x_1, x_2, \ldots, x_N)^\mathrm{T}$ and the interested signal consisting of independent components (ICs) be $\mathbf{c} = (c_1, c_2, \ldots, c_M)^\mathrm{T}$, and generally $M \leq N$. The signal $\mathbf{x}$ is considered to be a linear mixture of independent components $\mathbf{c}$: $\mathbf{x} = \mathbf{A}\mathbf{c}$, where $\mathbf{A}$ is an $N \times M$ mixing matrix with full column rank.

The goal of general ICA is to obtain a linear $M \times N$ demixing matrix $\mathbf{W}$ to recover the independent components $\mathbf{c}$ with a minimal knowledge of $\mathbf{A}$ and $\mathbf{c}$, normally $M = N$. Then, the recovered components $\mathbf{u}$ are given by $\mathbf{u} = \mathbf{W}\mathbf{x}$.

In the present paper, the contrast function used is the mutual information ($\mathcal{M}$) of the output signal which is defined in the sense of variable's entropy to measure the independence:

$$\mathcal{M}(\mathbf{W}) = \sum_{i=1}^{M} H(u_i) - H(\mathbf{u}) \tag{4}$$

where $H(u_i)$ is the marginal entropy of component $u_i$ and $H(\mathbf{u})$ is the output joint entropy. $\mathcal{M}$ has non-negative value and equals to zero when components are completely independent.

While minimizing $\mathcal{M}$, the learning equation for demixing matrix $\mathbf{W}$ to perform ICA is given by [1]:

$$\Delta\mathbf{W} \propto \mathbf{W}^{-\mathrm{T}} + \mathbf{\Phi}(\mathbf{u})\mathbf{x}^\mathrm{T} \tag{5}$$

where $\mathbf{W}^{-\mathrm{T}}$ is the transpose of the inverse matrix $\mathbf{W}^{-1}$ and $\mathbf{\Phi}(\mathbf{u})$ is a nonlinear function depending on the activation functions of neurons or p.d.f. of sources [1]. With above assumptions, the exact components $\mathbf{c}$ are indeterminant because of possible dilation and permutation. The independent components and the columns of $\mathbf{A}$ and the rows of $\mathbf{W}$ can only be estimated up to a multiplicative constant. The definitions of normal ICA imply no ordering of independent components [5].

## 4 Constrained ICA

In practice, the ordering of independent components is quite important to separate non-stationary signals or interested signals with significant statistical characters. Eliminating indeterminacy in the permutation and dilation is useful to produce a unique ICA solution with systematically ordered signals and normalized demixing matrix. This section presents an approach to CICA by enhancing classical ICA procedure using Lagrange multiplier methods to obtain unique ICs.

### 4.1 Ordering of Independent Components

The independent components are ordered in a descent manner according to a certain statistical measure defined as index $\mathcal{I}(\mathbf{u})$. The constrained optimization problem to CICA is then defined as follows:

$$\begin{aligned} \text{minimize} \qquad & \text{Mutual Information } \mathcal{M}(\mathbf{W}) \\ \text{subject to} \quad & \mathbf{g}(\mathbf{W}) \leq \mathbf{0}, \; \mathbf{g}(\mathbf{W}) = [g_1(\mathbf{W}) \cdots g_{M-1}(\mathbf{W})]^\mathrm{T} \end{aligned} \tag{6}$$

where $\mathbf{g}(\mathbf{W})$ is a set of $(M-1)$ inequality constraints, $g_i(\mathbf{W}) = \mathcal{I}(u_{i+1}) - \mathcal{I}(u_i)$ defines the descent order and $\mathcal{I}(u_i)$ is the index of some statistical measures of output components $u_i$, e.g. variance, normalized kurtosis.

Using Lagrange multiplier methods, the augmented Lagrangian function is defined

based on eq.(2) as:

$$\mathcal{L}(\mathbf{W},\mu) = \mathcal{M}(\mathbf{W}) + \frac{1}{2\gamma}\sum_{i=1}^{M-1}\{[\max\{0,\bar{g}_i(\mathbf{W})\}]^2 - \mu_i^2\} \tag{7}$$

With discrete solutions applied, the changes of individual element $w_{ij}$ can be formulated by minimizing eq.(7):

$$\Delta w_{ij} \propto \Delta_{w_{ij}}\mathcal{L}(\mathbf{W}(k),\mu(k)) = \min_{w_{ij}}\mathcal{M}(\mathbf{W}(k)) + [\max\{0,\bar{g}_{i-1}(\mathbf{W}(k))\}$$
$$- \max\{0,\bar{g}_i(\mathbf{W}(k))\}]\,\mathcal{I}'(u_i(k))\,x_j \tag{8}$$

where $\mathcal{I}'(\cdot)$ is the first derivative of index measure.

The iterative equation for finding individual multipliers $\mu_i$ is

$$\mu_i(k+1) = \max\{0,\mu_i(k) + \gamma\,[\mathcal{I}(u_{i+1}(k)) - \mathcal{I}(u_i(k))]\} \tag{9}$$

With the learning equation of normal ICNN given in (5) and the multiplier $\mu_i$'s iterative equation (9), the iterative procedure to determine the demixing matrix $\mathbf{W}$ is given as follows:

$$\Delta\mathbf{W} \propto \Delta_{\mathbf{W}}\mathcal{L}(\mathbf{W},\mu) = \mathbf{W}^{-T} + \Psi(\mathbf{u})\mathbf{x}^T \tag{10}$$

$$\text{where } \Psi(\mathbf{u}) = \begin{bmatrix} \Phi_1(u_1) - \mu_1\mathcal{I}'(u_1) \\ \Phi_2(u_2) + (\mu_1-\mu_2)\mathcal{I}'(u_2) \\ \vdots \\ \Phi_{M-1}(u_{M-1}) + (\mu_{M-2}-\mu_{M-1})\mathcal{I}'(u_{M-1}) \\ \Phi_M(u_M) + \mu_{M-1}\mathcal{I}'(u_M) \end{bmatrix}$$

We apply measures of variance and kurtosis as examples to emerge the ordering among the signals. Then the functions $\mathcal{I}$ and corresponding first-derivative $\mathcal{I}'$ become as below.

$$\text{variance}: \quad \mathcal{I}_{\text{var}}(u_i) = E\{u_i^T u_i\} \quad \mathcal{I}'_{\text{var}}(u_i) = 2E\{u_i\} \tag{11}$$

$$\text{kurtosis}: \quad \mathcal{I}_{\text{kur}}(u_i) = \frac{E\{u_i^4\}}{E\{u_i^2\}^2} - 3 \quad \mathcal{I}'_{\text{kur}}(u_i) = \frac{4u_i^3}{E\{u_i^2\}^2} - \frac{4E\{u_i^4\}u_i}{E\{u_i^2\}^3} \tag{12}$$

The signal with the most variance shows the majority of information that input signals consist of. The ordering based on variance sorts the components in information magnitude that needs to reconstruct the original signals. However, it should be used accompanying with other preprocessing or constraints, such as PCA or normalization, because the normal ICA's indeterminacy on dilation of demixing matrix may cause the variance of output components being amplified or reduced.

Normalized kurtosis is a kind of 4th-order statistical measure. The kurtosis of a stationary signal to be extracted is constant under the situation of indeterminacy on signals' amplitudes. Kurtosis shows the high order statistical character. Any signal can be categorized into super-Gaussian, Gaussian and sub-Gaussianly distributed ones by using kurtosis. The components are ordered in the distribution of sparseness (i.e. super-Gaussian) to denseness (i.e. sub-Gaussian). Kurtosis has been widely used to produce one-unit ICA [7]. In contrast to their sequential extraction, our approach can extract and order the components in parallel.

## 4.2  Normalization of Demixing Matrix

The definition of ICA implies an indeterminacy in the norm of the mixing and demixing matrix, which is in contrast to, e.g. PCA. Rather than the unknown

mixing matrix $\mathbf{A}$ was to be estimated, the rows of the demixing matrix $\mathbf{W}$ can be normalized by applying a constraint term in the ICA energy function to establish a normalized demixing channel. The constrained ICA problem is then defined as follows:

$$\text{minimize} \qquad \text{Mutual Information } \mathcal{M}(\mathbf{W})$$
$$\text{subject to} \quad \mathbf{h}(\mathbf{W}) = [h_1(\mathbf{W}) \cdots h_M(\mathbf{W})]^{\mathrm{T}} = \mathbf{0} \qquad (13)$$

where $\mathbf{h}(\mathbf{W})$ defines a set of $M$ equality constraints, $h_i(\mathbf{w}_i) = \mathbf{w}_i^{\mathrm{T}}\mathbf{w}_i - 1$ ($i = 1, \cdots, M$), which define the row norms of the demixing matrix $\mathbf{W}$ equal to 1.

Using Lagrange multiplier methods, the augmented Lagrangian function is defined based on eq.(2) as:

$$\mathcal{L}(\mathbf{W}, \lambda) = \mathcal{M}(\mathbf{W}) + \lambda^{\mathrm{T}}\text{diag}[\mathbf{W}\mathbf{W}^{\mathrm{T}} - \mathbf{I}] + \frac{1}{2}\gamma \, \|\text{diag}[\mathbf{W}\mathbf{W}^{\mathrm{T}} - \mathbf{I}]\|^2 \qquad (14)$$

where $\text{diag}[\cdot]$ denotes the operation to select the diagonal elements in the square matrix as a vector.

By applying discrete Lagrange multiplier method, the iterative equation minimizing the augmented function for individual multiplier $\lambda_i$ is

$$\lambda_i(k+1) = \lambda_i(k) + \gamma \left(\mathbf{w}_i^{\mathrm{T}}\mathbf{w}_i - 1\right) \qquad (15)$$

and the iterative equation of demixing matrix $\mathbf{W}$ is given as follows:

$$\Delta\mathbf{W} \propto \Delta_{\mathbf{W}}\mathcal{L}(\mathbf{W}, \lambda) = \mathbf{W}^{-\mathrm{T}} + \mathbf{\Phi}(\mathbf{u})\mathbf{x}^{\mathrm{T}} + \mathbf{\Omega}(\mathbf{W}) \qquad (16)$$
$$\text{where } \Omega_i(w_i) = 2\lambda_i w_i^{\mathrm{T}}$$

Let assume $\mathbf{c}$ is the normalized source with unit variance such that $E\{\mathbf{c}\mathbf{c}^{\mathrm{T}}\} = \mathbf{I}$, and the input signal $\mathbf{x}$ is processed by a prewhitening matrix $\mathbf{P}$ such that $\mathbf{p} = \mathbf{P}\mathbf{x}$ obeys $E\{\mathbf{p}\mathbf{p}^{\mathrm{T}}\} = \mathbf{I}$. Then with the normalized demixing matrix $\mathbf{W}$, the network output $\mathbf{u}$ contains exact independent components with unit magnitude, i.e. $u_i$ contains one $\pm c_j$ for some non-duplicative assignment $j \to i$.

## 5 Experiments and Results

The CICA algorithms were simulated in MATLAB version 5. The learning procedure ran 500 iterations with certain learning rate. All signals were preprocessed by a whitening process to have zero mean and uniform variance. The accuracy of the recovered components compared to the source components was measured by the signal to noise ratio (SNR) in dB, where signal power was measured by the variance of the source component, and noise was the mean square error between the sources and recovered ones. The performance of the network separating the signals into ICs was measured by an individual performance index (IPI) of the permutation error $\epsilon_i$ for $i$th output:

$$\epsilon_i = \left(\sum_{j=1}^{n} \frac{|p_{ij}|}{\max_k |p_{ik}|}\right) - 1 \qquad (17)$$

where $p_{ij}$ were elements of the permutation matrix $\mathbf{P} = \mathbf{W}\mathbf{A}$. IPI was close to zero when the corresponding output was closely independent to other components.

### 5.1 Ordering ICs in Signal Separation

Three independent random signals distributed in Gaussian, sub- and super-Gaussian manner were simulated. Their statistical configurations were similar to those used

in [1]. These source signals **c** were mixed with a random matrix to derive inputs to the network. The networks were trained to obtain the $3 \times 3$ demixing matrix using the algorithm of kurtosis-constraint CICA eq.(10) and (12) to separate three independent components in complete ICA manner.

The source components, mixed input signals and the resulted output waveforms are shown in figure 1 (a), (b) and (c), respectively. The network separated and

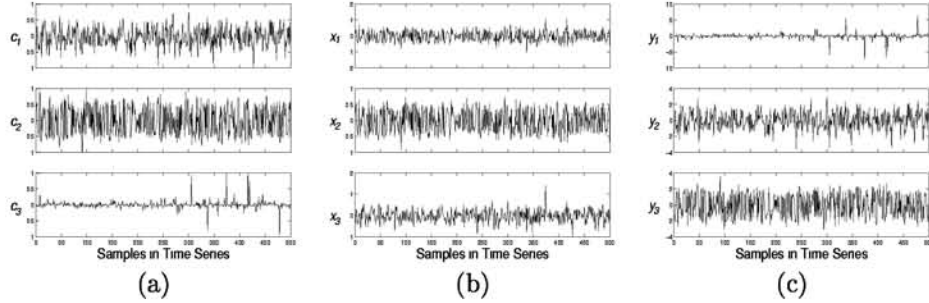

|  | (a) | (b) | (c) |

Figure 1: Result of extraction of one super-Gaussian, one Gaussian and one sub-Gaussian signals in the kurtosis descent order. Normalized kurtosis measurements are $\kappa_4(y_1) = 32.82$, $\kappa_4(y_2) = -0.02$ and $\kappa_4(y_3) = -1.27$. (a) Source components, (b) input mixtures and (c) resulted components.

sorted the output components in a decreasing manner of kurtosis values, where the component $y_1$ had kurtosis 32.82 ($> 0$, super-Gaussian), $y_2$ is 0.02 ($\approx 0$, Gaussian) and $y_3$ is -1.27 ($< 0$, sub-Gaussian). The final performance index value of 0.28 and output components' average SNR value of 15dB show all three independent components well separated too.

## 5.2 Demixing Matrix Normalization

Three deterministic signals and one Gaussian noise were simulated in this experiment. All signals were independently generated with unit variance and mixed with a random mixing matrix. All input mixtures were preprocessed by a whitening process to have zero mean and unit variance. The signals were separated using both unconstrained ICA and constrained ICA as given by eq.(5) and (16) respectively.

Table 1 compares their resulted demixing matrix, row norms, variances of separated components and SNR values. The dilation effect can be seen from the difference

| | y | Demixing Matrix **W** | | | | Norms | Variance | SNR |
|---|---|---|---|---|---|---|---|---|
| | $y_1$ | 0.90 | 0.08 | −0.12 | −0.82 | 1.23 | 1.50 | 4.55 |
| uncons. | $y_2$ | −0.06 | 1.11 | −0.07 | 0.07 | 1.11 | 1.24 | 10.88 |
| ICA | $y_3$ | 0.07 | 0.07 | 1.47 | −0.09 | 1.47 | 2.17 | 21.58 |
| | $y_4$ | 1.04 | 0.08 | 0.04 | 1.16 | 1.56 | 2.43 | 16.60 |
| | $y_1$ | 0.65 | 0.43 | −0.02 | −0.61 | 0.99 | 0.98 | 4.95 |
| cons. | $y_2$ | −0.37 | 0.91 | 0.05 | 0.20 | 1.01 | 1.02 | 13.94 |
| ICA | $y_3$ | 0.01 | −0.04 | 1.00 | −0.04 | 1.00 | 1.00 | 25.04 |
| | $y_4$ | 0.65 | 0.07 | 0.02 | 0.76 | 1.00 | 1.00 | 22.56 |

Table 1: Comparison of the demixing matrix elements, row norms, output variances and resulted components' SNR values in ICA, and CICA with normalization.

among components' variances caused by the non-normalized demixing matrix in unconstrained ICA. The CICA algorithm with normalization constraint normalized rows of the demixing matrix and separated the components with variances remained at unit. Therefore, the source signals are exactly recovered without any dilation. The increment of separated components' SNR values using CICA also can be seen in the table. Their source components, input mixture, separated components using normalization are given in figure 2. It shows that the resulted signals from CICA are exactly match with the source signals in the sense of waveforms and amplitudes.

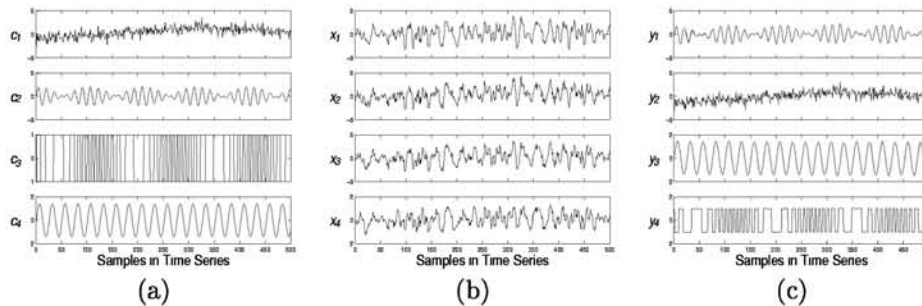

Figure 2: (a) Four source deterministic components with unit variances, (b) mixture inputs and (c) resulted components through normalized demixing channel $\mathbf{W}$.

## 6   Conclusion

We present an approach of constrained ICA using Lagrange multiplier methods to eliminate the indeterminacy of permutation and dilation which are present in classical ICA. Our results provide a technique for systematically enhancing the ICA's usability and performance using the constraints not restricted to the conditions treated in this paper. More useful constraints can be considered in similar manners to further improve the outputs of ICA in other practical applications. Simulation results demonstrate the accuracy and the usefulness of the proposed algorithms.

## References

[1] Jagath C. Rajapakse and Wei Lu. Unified approach to independent component networks. In *Second International ICSC Symposium on NEURAL COMPUTATION (NC'2000)*, 2000.

[2] A. Bell and T. Sejnowski. An information-maximization approach to blind separation and blind deconvolution. *Neurocomputing*, 7:1129–1159, 1995.

[3] S. Amari, A. Chchocki, and H. Yang. A new learning algorithm for blind signal separation. In *Advances in Neural Information Processing Systems 8*, 1996.

[4] T-W. Lee, M. Girolami, and T. Sejnowski. Independent component analysis using an extended informax algorithm for mixed sub-gaussian and super-gaussian sources. *Neural Computation*, 11(2):409–433, 1999.

[5] P. Comon. Independent component analysis: A new concept? *Signal Processing*, 36:287–314, 1994.

[6] Dimitri P. Bertsekas. *Constrained optimization and Lagrange multiplier methods*. New York : Academic Press, 1982.

[7] A. Hyvärinen and Erkki Oja. Simple neuron models for independent component analysis. *Neural Systems*, 7(6):671–687, December 1996.
